# Robust Parameter Estimation And Model Selection For Neural Network Regression

**Yong Liu**
Department of Physics
Institute for Brain and Neural Systems
Box 1843, Brown University
Providence, RI 02912
yong@cns.brown.edu

## Abstract

In this paper, it is shown that the conventional back-propagation (BPP) algorithm for neural network regression is robust to leverages (data with $x$ corrupted), but not to outliers (data with $y$ corrupted). A robust model is to model the error as a mixture of normal distribution. The influence function for this mixture model is calculated and the condition for the model to be robust to outliers is given. EM algorithm [5] is used to estimate the parameter. The usefulness of model selection criteria is also discussed. Illustrative simulations are performed.

## 1 Introduction

In neural network research, the back-propagation (BPP) algorithm is the most popular algorithm. In the regression problem $y = \eta(x, w) + \varepsilon$, in which $\eta(x, \theta)$ denote a neural network with weight $\theta$, the algorithm is equivalent to modeling the error as identically independently normally distributed (i.i.d.), and using the maximum likelihood method to estimate the parameter [13]. Howerer, the training data set may contain *surprising* data points either due to errors in $y$ space (outliers) when the response vectors $y$s of these data points are far away from the underlying function surface, or due to *errors* in $x$ space (leverages), when the the feature vectors

$x$s of these data points are far away from the mass of the feature vectors of the rest of the data points. These abnormal data points may be able to cause the parameter estimation biased towards them. A robust algorithm or robust model is the one that overcome the influence of the abnormal data points.

A lot of work has been done in linear robust regression [8, 6, 3]. In neural network. it is generally believed that the role of sigmoidal function of the basic computing unit in the neural net has some significance in the robustness of the neural net to outliers and leverages. In this article, we investigate this more thoroughly. It turns out the conventional normal model (BPP algorithm) is robust to leverages due to sigmoidal property of the neurons, but not to outliers (section 2). From the Bayesian point of view [2], modeling the error as a mixture of normal distributions with different variances, with flat prior distribution on the variances, is more robust. The influence function for this mixture model is calculated and condition for the model to be robust to outliers is given (section 3.1). An efficient algorithm for parameter estimation in this situation is the EM algorithm [5] (section 3.2). In section 3.3, we discuss a choice of prior and its properties. In order to choose among different probability models or different forms of priors, and neural nets with different architecture, we discuss the model selection criteria in section 4. Illustrative simulations on the choice of prior, or the $t$ distribution model, and the normal distribution model are given. Model selection statistics, is used to choose the degree of freedom of $t$ distribution, different neural network, and choose between a $t$ model and a normal model (section 4 and 5).

## 2  Issue Of Robustness In Normal Model For Neural Net Regression

One way to think of the outliers and leverages is to regard them as a data perturbation on the data distribution of the good data. Remember that a estimated parameter $T = T(F)$ is an implicit function of the underlying data distribution $F$. To evaluate the influence of $T$ by this distribution perturbation, we use the influence function [6] of estimator $T$ at point $z = (x, y)$ with data distribution F, which is defined as

$$\text{IF}(T, z, F) = \lim_{t \to 0+} \frac{T((1-t)F + t\Delta_z) - T(F)}{t} \tag{1}$$

in which $\Delta_x$ has mass 1 at $x$.[1] This definition is equivalent to a definition of derivative with respect to $F$ except what we are dealing now is the derivative of functional. This definition gives the amount of change in the estimator $T$ with respect to a distribution perturbation $t\Delta_z$ at point $z = (x, y)$. For a robust estimation of the parameter, we expect the estimated parameter does not change significantly with respect to a data perturbation. In another word, the influence function is bounded for a robust estimation.

Denote the conditional probability model of $y$ given $x$ as i.i.d. $f(y|x, \theta)$ with parameter $\theta$. If the error function is the negative log-likelihood plus or not plus a penalty term, then a general property of the influence function of the estimated parameter $\hat{\theta}$, is $\text{IF}(\hat{\theta}, (x_i, y_i), F) \propto \nabla_\theta \log f(y_i|x_i, \hat{\theta})$ (for proof, see [11]). Denote the neural

net, with $h$ hidden units and the dimension of the output being one ($d_y = 1$), as

$$\eta(x, \theta) = \sum_{k=1}^{h} a_k \sigma(w_k x + t_k) \qquad (2)$$

in which $\sigma(x)$ is the sigmoidal function or $1/(1 + \exp(x))$ and $\theta = \{a_k, w_k, t_k\}$. For a normal model, $f(y|x, \theta, \sigma) = \mathcal{N}(y; \eta(x, \theta), \sigma)$ in which $\mathcal{N}(y; c, \sigma)$ denotes $d_y$-variate normal distribution with mean $c$ and covariance matrix $\sigma^2 I$. Straightforward calculation yield ($d_y = 1$)

$$\text{IF}(\hat{\theta}, (x_i, y_i), F) \propto (y - \eta(x, \hat{\theta})) \left( \begin{array}{c} (\sigma(\hat{w}_i x + \hat{t}_i))_{h \times 1} \\ \left( \begin{array}{c} \hat{a}_i \sigma'(\hat{w}_i x + \hat{t}_i) x \\ \hat{a}_i \sigma'(\hat{w}_i x + \hat{t}_i) \end{array} \right)_{h \times 1} \end{array} \right) \qquad (3)$$

Since $y$ with a large value makes the influence function unbounded, thus the normal model or the back-propagation algorithm for regression is not robust to outliers. Since $\sigma'(wx + t)$ tends to be zero for $x$ that is far away from the projection $\hat{w}x + \hat{t} = 0$, the influence function is bounded for a abnormal $x$, or the normal model for regression is robust to leverages. This analysis can be easily extented to a neural net with multiple hidden layers and multiple outputs. Since the neural net model is robust to leverages, we shall concentrate on the discussion of robustness with respect to outliers afterwards.

## 3   Robust Probability Model And Parameter Estimation

### 3.1   Mixture Model

One method for the robust estimation is by the Bayesian analysis [2]. Since our goal is to overcome the influence of outliers in the data set, we model the error as a mixture of normal distributions, or,

$$f(y|x, \theta, \sigma) = \int f(y|x, \theta, q, \sigma) \pi(q) dq \qquad (4)$$

with $f(y|x, \theta, q, \sigma) = \mathcal{N}(y; \eta(x, \theta), \sigma^2/q)$ and the prior distribution on $q$ is denoted as $\pi(q)$. Intuitively, a mixture of different normal distributions with different $q$s, or different variances, somehow conveys the idea that a data point is generated from a normal distribution with large variance, which can be considered to be outliers, or from that with small variance, which can be considered to be good data. This requires $\pi(q)$ to be flat to accommodate the abnormal data points. A case of extreme non-flat prior is to choose $\pi(q) = \delta(q - 1)$, which will make $f(y|x, \theta, \sigma)$ to be a normal distribution model. This model has been discussed in previous section and it is not robust to outliers.

Calculation yields ($d_y = 1$) the influence function as

$$\text{IF}(\hat{\theta}, (x, y), F) \propto (y - \eta(x, \hat{\theta})) \, \hat{w} \left( \begin{array}{c} (\sigma(\hat{w}_i x + \hat{t}_i))_{h \times 1} \\ \left( \begin{array}{c} \hat{a}_i \sigma'(\hat{w}_i x + \hat{t}_i) x \\ \hat{a}_i \sigma'(\hat{w}_i x + \hat{t}_i) \end{array} \right)_{h \times 1} \end{array} \right) \qquad (5)$$

in which

$$\hat{w} = \mathrm{E}\left[q|y, x, \sigma, \hat{\theta}\right] \qquad (6)$$

where expectation is taken with respect to the posterior distribution of $q$, or $\pi(q|y, x, \sigma, \theta) = \frac{f(y|x,\theta,q,\sigma)\pi(q)}{f(y|x,\theta,\sigma)}$ For the influence function to be bounded for a $y$ with large value, $(y - \eta(x, \hat{\theta}))\hat{w}$ must be bounded. This is the condition on $\pi(q)$ when the distribution $f(y|x, \theta, \sigma)$ is robust to outliers. It can be noticed that the mixture model is robust to leverages for the same reason as in the case of the normal distribution model.

## 3.2    Algorithm For Parameter Estimation

An efficient parameter estimation method for the model in equation 4 is the EM algorithm [5]. In EM algorithm, a portion of the parameter is regarded as the missing observations. During the estimation, these missing observations are estimated through previous estimated parameter of the model. Afterwards, these estimated missing observations are combined with the real observations to estimate the parameter of the model. In our mixture model, we shall regard $\{q_i, i = 1, ...n\}$ as the missing observations. Denote $\omega = \{x_i, y_i, i = 1, ...n\}$ as the training data set.

It is a straight forward calculation for the EM algorithm (see Liu, 1993b) once one write down the full probability $f(\{y_i, q_i\}|\{x_i\}, \sigma, \theta)$. The algorithm is equivalent to minimizing

$$\sum_{i=1}^{n} w_i^{(s-1)}(y_i - \eta(x_i, \theta))^2 \qquad (7)$$

and estimating $\sigma$ at the $s$ step by $(\hat{\sigma}^2)^{(s)} = \frac{1}{n}\sum_{i=1}^{n} w_i^{(s-1)}(y_i - \eta(x_i, \hat{\theta}^{(s)}))^2$.

If we use $f(y|x, \theta, \sigma) \propto \exp(-\rho(|y-\eta(x, \theta)|/\sigma))$ and denote $\psi(z) = \rho'(z)$, calculation yield, $w = \mathrm{E}\left[q|y, x, \sigma, \theta\right] = \frac{\psi(z)}{z} |_{z=|y-\eta(x,\theta)|/\sigma}$. This has exact the same choice of weight $w_i^{(s-1)}$ as in the iterative reweighted regression algorithm [7]. What we have here, different from the work of Holland et al., is that we link the EM algorithm with the iterative reweighted algorithm, and also extend the algorithm to a much more general situation. The weighting $w_i$ provides a measure of the goodness of a data point. Equation 7 estimates the parameters based on the portion of the data that are good. A penalization term on $\theta$ can also be included in equation 7. [2]

## 3.3    Choice Of Prior

There are a lot choices of prior distribution $\pi(q)$ (for discussion, see [11]). We only discuss the choice $\nu q \sim \chi_\nu^2$, i.e., a chi distribution with $\nu$ degree of freedom. By intergrating equation 4, $f(y|x, \theta, \sigma) = \frac{\Gamma((\nu+d_y)/2)}{\Gamma(\nu/2)(\sigma^2\nu\pi)^{d_y/2}}(1 + \frac{(y-\eta(x,\theta))^2}{\nu\sigma^2})^{-(\nu+d_y)/2}$. It is a $d_y$ variate $t$ distribution with $\nu$ degree of freedom, mean 0 and covariance matrix $\sigma^2 I$. Calculation yields, $\mathrm{E}\left[q|y, x, \sigma, \theta\right] = \frac{\nu+d_y}{\nu+(y-\eta(x,\theta))^2/\sigma^2}$ The $t$ distribution

becomes a normal distribution as $\nu$ goes to infinity. For finite $\nu$, it has heavier tail than the normal distribution and thus is appropriate for regression with to outliers. Actually the condition for robustness, $(y - \eta(x, \hat{\theta}))\hat{w}$ being bounded for a $y$ with large value, is satisfied. The weighting $w \propto 1/\{1 + \left[y - \eta(x, \hat{\theta})\right]^2 / \hat{\sigma}^2\}$ balances the influence of the $y$s with large values, achieving robustness with respect to outliers for the $t$ distribution.

## 4  Model Selection Criteria

The meaning of model is in a broad sense. It can be the degree of penalization, or a probability model, or a neural net architecture, or the combination of the above. A lot of work has been done in model selection [1, 17, 15, 4, 13, 14, 10, 12]. The choice of a model is based on its prediction ability. A natural choice is the expected negative log-likelihood. This is equivalent to using the Kullback-Leibler measure [9] for model selection, or $-\mathrm{E}\left[\log f(y|x, \text{model})\right] + \mathrm{E}\left[\log f(y|x, \text{true model})\right]$. This has problem if the model can not be normalized as in the case of a improper prior. Equation 7 implies that we can use

$$T_m(\omega) = \frac{1}{n_{eff}} \sum_{i=1}^{n} w_i^* \left(y_i - \eta(x_i, \hat{\theta}_{-i})\right)^2 \qquad (8)$$

as the cross-validation [16] assessment of model $m$, in which $n_{eff} = \sum_i w_i^*$, $w_i^*$ is the convergence limit of $w_i^{(s)}$, or equation 6, and $\hat{\theta}_{-i}$ is the estimator of $\theta$ with $i$th data deleted from the full data set. The successfulness of the cross-validation method depends on a robust parameter estimation. The cross-validation method is to calculate the average prediction error on a data based on the rest of the data in the training data set. In the presence of outliers, predicting an outlier based on the rest of the data, is simply not meaningful in the evaluation of the model. Equation 8 takes consideration of the outliers. Using result from [10], we can show [11] with penalization term $\alpha(\lambda, \theta)$,

$$T_m(\omega) \approx \frac{1}{n_{eff}} \sum_{i=1}^{n} w_i^* \left(y_i - \eta(x_i, \hat{\theta})\right)^2 \qquad (9)$$

$$+ \frac{1}{n_{eff}} \sum_{i=1}^{n} w_i^* r_i g_i^t \left[\sum_i w_i^* (g_i g_i^t - r_i \zeta_i) + \nabla_\theta \nabla_\theta \alpha(\lambda, \theta)\right]^{-1} r_i g_i \quad (10)$$

in which $g_i = \nabla_\theta \eta(x_i, \hat{\theta})$, $\zeta_i = \nabla_\theta \nabla_\theta^t \eta(x_i, \hat{\theta})$ and $r_i = y_i - \eta(x_i, \hat{\theta})$. Thus if the models in comparison contains a improper prior, the above model selection statistics can be used.

If the models in comparison have close forms of $f(y|x, \theta, \sigma)$, the average negative log-likelihood can be used as the model selection criteria. In Liu's work [10], an approximation form for the unbiased estimation of expected negative log-likelihood was provided. If we use the negative log-likelihood plus a penalty term $\alpha(\lambda, \theta)$ as the parameter optimization criteria, the model selection statistics is

$$T_m(\omega) = -\frac{1}{n} \sum_{i=1}^{n} \log f(y_i|x_i, \hat{\theta}_{-i}) \approx -\frac{1}{n} \sum_{i=1}^{n} \log f(y_i|x_i, \hat{\theta}) + \frac{1}{n} \mathrm{Tr}(C^{-1}D) \quad (11)$$

in which $C = \sum_{i=1}^{n} \nabla_{\theta} \log f(y_i|x_i, \hat{\theta}) \nabla_{\theta}^{t} \log f(y_i|x_i, \hat{\theta})$ and $D = -\sum_{i=1}^{n} \nabla_{\theta} \nabla_{\theta}^{t} \log f(y_i|x_i, \hat{\theta}) + \nabla_{\theta} \nabla_{\theta}^{t} \alpha(\lambda, \theta)$. The optimal model is the one that minimizes this statistics. If the true underlying distribution is the normal distribution model and there is no penalization terms, it is easy to prove $C \to D$ as $n$ goes to infinite. Then the statistics becomes AIC [1].

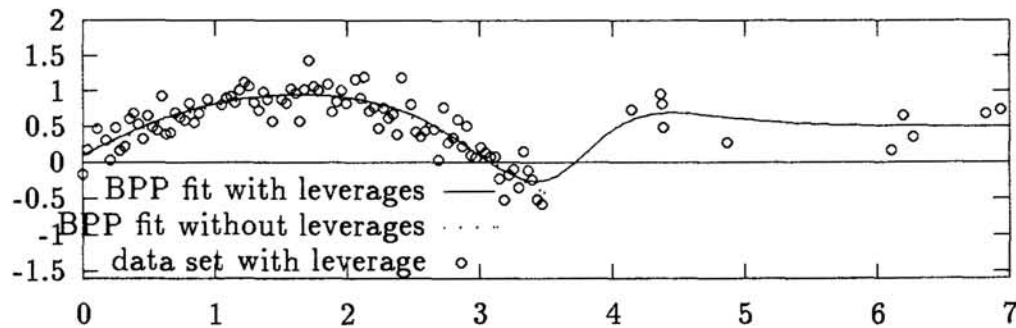

Figure 1: BPP fit to data set with leverages, and comparison with BPP fit to the data set without the leverages. An one hidden layer neural net with 4 hidden units, is fitted to a data set with 10 leverage, which are on the right side of $x = 3.5$, by using the conventional BPP method. The main body of the data (90 data points) was generated from $y = \sin(x) + \epsilon$ with $\epsilon \sim \mathcal{N}(\epsilon; 0, \sigma = 0.2)$. It can be noticed that the fit on the part of good data points was not dramatically influenced by the leverages. This verified our theoretical result about the robustness of a neural net with respect to leverages

## 5    Illustrative Simulations

For the results shown in figure 2 and 3, the training data set contains 93 data point from $y = \sin(x) + \epsilon$ and seven $y$ values (outliers) randomly generated from region $[1, 2)$, in which $\epsilon \sim \mathcal{N}(\epsilon; 0, \sigma = 0.2)$. The neural net we use is of the form in equation 2. Denote $h$ as the number of hidden units in the neural net. The caption of each figure (1, 2, 3) explains the usefulness of the parameter estimation algorithm and the model selection.

### Acknowledgements

The author thanks Leon N Cooper, M. P. Perrone. The author also thanks his wife Cong. This research was supported by grants from NSF, ONR and ARO.

## Footnotes

[1]The probability density of the distribution $\Delta_x$ is $\delta(y - x)$.

[2]A prior on $\theta$ can be $\pi(\theta) \propto e^{-\alpha(\lambda,\theta)/(2\sigma^2)}$, which yields a additional penalization term $\alpha(\lambda, \theta)$ in equation 7, in which $\lambda$ denotes a tunning parameters of the penalization.

## References

[1] H. Akaike. Information theory and an extension of the maximum likelihood

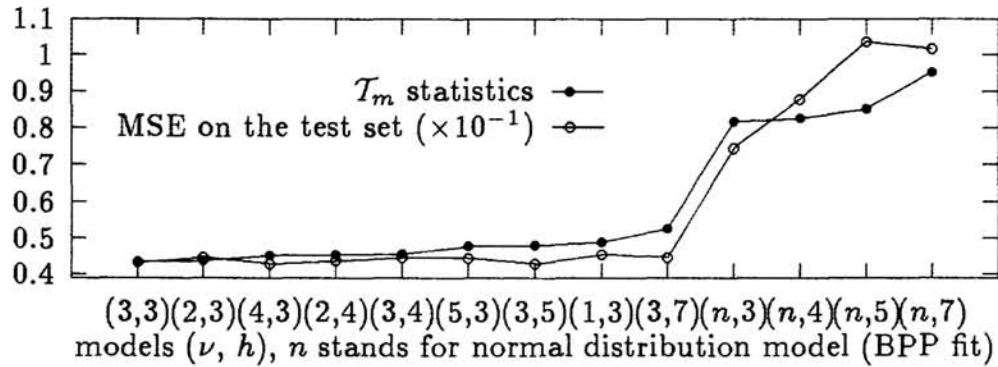

Figure 2: Model selection statistics $\mathcal{T}_m$ for fits to data set with outliers, tests on a independent data set with 1000 data points from $y = \sin(x) + \epsilon$, where $\epsilon \sim \mathcal{N}(\epsilon; 0, \sigma = 0.2)$. it can be seen that $\mathcal{T}_m$ statistics is in consistent with the error on the test data set. The $\mathcal{T}_m$ statistics favors $t$ model with small $\nu$ than for the normal distribution models.

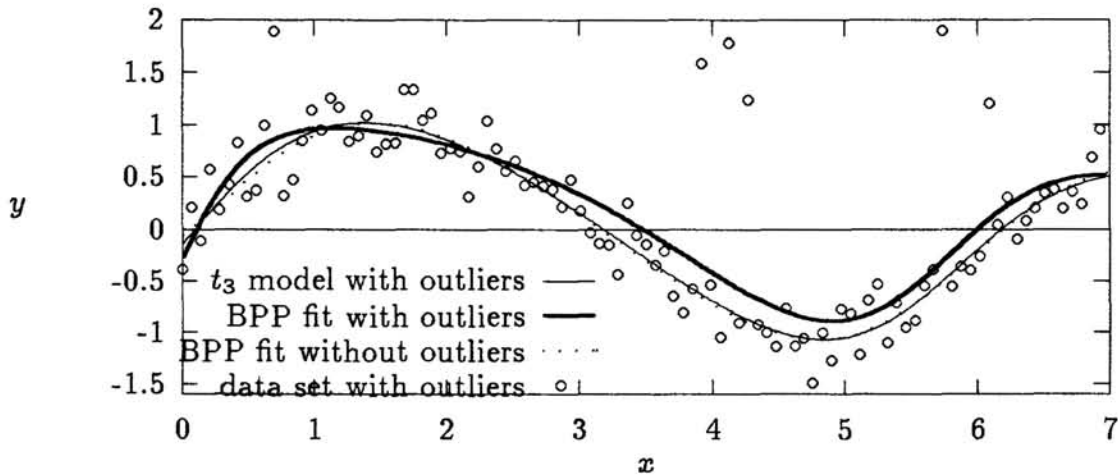

Figure 3: Fits to data set with outliers, and comparison with BPP fit to the data set without the outliers. The best fit in the four BPP fits ($h = 3$), according to $\mathcal{T}_m$ statistics, was influenced by the outliers, tending to shift upwards. Although the distribution is not a $t$ distribution at all, the best fit by the EM algorithm under the $t$ model ($\nu = 3, h = 3$), also according to $\mathcal{T}_m$ statistics, gives better result than the BPP fit, actually is almost the same as the BPP fit ($h = 3$) to the training data set without the outliers. This is due to the fact that a $t$ distribution has a heavy tail to accommodate the outliers

principle. In Petrov and Czaki, editors, *Proceedings of the 2nd International Symposium on Information Theory*, pages 267–281, 1973.

[2] J. O. Berger. *Statistical Decision Theory and Bayesian Analysis.* Springer-Verlag, 1985.

[3] R. D. Cook and S. Weisberg. Characterization of an empirical influence function for detecting influential cases in regression. *Technometrics*, 22:495–508, 1980.

[4] P. Craven and G. Wahba. Smoothing noisy data with spline functions:estimating the correct degree of smoothing by the method of generalized cross-validation. *Numer. Math.*, 31:377–403, 1979.

[5] A. P. Dempster, N. M. Laird, and D. B. Rubin. Maximum likelihood from incomplete data via the EM algorithm. (with discussion). *J. Roy. Stat. Soc. Ser. B*, 39:1–38, 1977.

[6] F.R. Hampel, E.M. Rouchetti, P.J. Rousseeuw, and W.A. Stahel. *Robust Statistics: The approach based on influence functions.* Wiley, 1986.

[7] P.J. Holland and R.E. Welsch. Robust regression using iteratively reweighted least-squares. *Commun. Stat. A*, 6:813–88, 1977.

[8] P.J. Huber. *Robust Statistics.* New York: Wiley, 1981.

[9] S. Kullback and R.A. Leibler. On information and sufficiency. *Ann. Stat.*, 22:79–86, 1951.

[10] Y. Liu. Neural Network Model Selection Using Asymptotic Jackknife Estimator and Cross-Validation Method. In C.L. Giles, S.J.and Hanson, and J.D. Cowan, editors, *Advances in neural information processing system 5*. Morgan Kaufmann Publication, 1993.

[11] Y. Liu. Robust neural network parameter estimation and model selection for regression. *Submitted.*, 1993.

[12] Y. Liu. Unbiased estimate of generalization error and model selection criterion in neural network. *Submitted to Neural Network*, 1993.

[13] D. MacKay. *Bayesian methods for adaptive models.* PhD thesis, California Institute of Technology, 1991.

[14] J. E. Moody. The effective number of parameters, an analysis of generalization and regularization in nonlinear learning system. In J. E. Moody, S. J. Hanson, and R. P. Lippmann, editors, *Advances in neural information processing system 4*, pages 847–854. Morgan Kaufmann Publication, 1992.

[15] G. Schwartz. Estimating the dimension of a model. *Ann. Stat*, 6:461–464, 1978.

[16] M. Stone. Cross-validatory choice and assessment of statistical predictions (with discussion). *J. Roy. Stat. Soc. Ser. B*, 36:111–147, 1974.

[17] M. Stone. An asymptotic equivalence of choice of model by cross-validation and Akaike's criterion. *J. Roy. Stat. Soc.*, Ser. B, 39(1):44–47, 1977.